# Independent Component Analysis of Electroencephalographic Data

**Scott Makeig**
Naval Health Research Center
P.O. Box 85122
San Diego CA 92186-5122
scott@cpl_mmag.nhrc.navy.mil

**Anthony J. Bell**
Computational Neurobiology Lab
The Salk Institute, P.O. Box 85800
San Diego, CA   92186-5800
tony@salk.edu

**Tzyy-Ping Jung**
Naval Health Research Center and
Computational Neurobiology Lab
The Salk Institute, P.O. Box 85800
San Diego, CA   92186-5800
jung@salk.edu

**Terrence J. Sejnowski**
Howard Hughes Medical Institute and
Computational Neurobiology Lab
The Salk Institute, P.O. Box 85800
San Diego, CA   92186-5800
terry@salk.edu

## Abstract

Because of the distance between the skull and brain and their different resistivities, *electroencephalographic* (EEG) data collected from any point on the human scalp includes activity generated within a large brain area. This spatial smearing of EEG data by volume conduction does not involve significant time delays, however, suggesting that the *Independent Component Analysis* (ICA) algorithm of Bell and Sejnowski [1] is suitable for performing blind source separation on EEG data. The ICA algorithm separates the problem of source identification from that of source localization. First results of applying the ICA algorithm to EEG and *event-related potential* (ERP) data collected during a sustained auditory detection task show: (1) ICA training is insensitive to different random seeds. (2) ICA may be used to segregate obvious artifactual EEG components (line and muscle noise, eye movements) from other sources. (3) ICA is capable of isolating overlapping EEG phenomena, including alpha and theta bursts and spatially-separable ERP components, to separate ICA channels. (4) Nonstationarities in EEG and behavioral state can be tracked using ICA via changes in the amount of residual correlation between ICA-filtered output channels.

# 1 Introduction

## 1.1 Separating What from Where in EEG Source Analysis

The joint problems of EEG source segregation, identification, and localization are very difficult, since the problem of determining brain electrical sources from potential patterns recorded on the scalp surface is mathematically underdetermined. Recent efforts to identify EEG sources have focused mostly on performing spatial segregation and localization of source activity [4]. By applying the ICA algorithm of Bell and Sejnowski [1], we attempt to completely separate the twin problems of source identification (What) and source localization (Where). The ICA algorithm derives independent sources from highly correlated EEG signals statistically and without regard to the physical location or configuration of the source generators. Rather than modeling the EEG as a unitary output of a multidimensional dynamical system, or as "the roar of the crowd" of independent microscopic generators, we suppose that the EEG is the output of a number of statistically independent but spatially fixed potential-generating systems which may either be spatially restricted or widely distributed.

## 1.2 Independent Component Analysis

Independent Component Analysis (ICA) [1, 3] is the name given to techniques for finding a matrix, $\mathbf{W}$ and a vector, $w$, so that the elements, $\mathbf{u} = [u_1 \ldots u_N]^T$, of the linear transform $\mathbf{u} = \mathbf{W}\mathbf{x} + w$ of the random vector, $\mathbf{x} = [x_1 \ldots x_N]^T$, are statistically independent. In contrast with decorrelation techniques such as Principal Components Analysis (PCA) which ensure that $\langle u_i u_j \rangle = 0, \forall ij$, ICA imposes the much stronger criterion that the multivariate probability density function (p.d.f.) of $\mathbf{u}$ factorizes: $f_{\mathbf{u}}(\mathbf{u}) = \prod_{i=1}^{N} f_{u_i}(u_i)$. Finding such a factorization involves making the mutual information between the $u_i$ go to zero: $I(u_i, u_j) = 0, \forall ij$. Mutual information is a measure which depends on all higher-order statistics of the $u_i$ while decorrelation only takes account of 2nd-order statistics.

In [1], a new algorithm was proposed for carrying out ICA. The only prior assumption is that the unknown independent components, $u_i$, each have the same form of cumulative density function (c.d.f.) after scaling and shifting, and that we know this form, call it $F_u(u)$. ICA can then be performed by maximizing the entropy, $H(\mathbf{y})$, of a non-linearly transformed vector: $\mathbf{y} = F_u(\mathbf{u})$. This yields stochastic gradient ascent rules for adjusting $\mathbf{W}$ and $w$:

$$\Delta \mathbf{W} \propto [\mathbf{W}^T]^{-1} + \hat{\mathbf{y}}\mathbf{x}^T, \Delta w \propto \hat{\mathbf{y}} \tag{1}$$

where $\hat{\mathbf{y}} = [\hat{y}_1 \ldots \hat{y}_N]^T$, the elements of which are:

$$\hat{y}_i = \frac{\partial}{\partial y_i} \frac{\partial y_i}{\partial u_i} \quad \text{[which if } \mathbf{y} = F_u(\mathbf{u})\text{]} \quad = \frac{\partial f_u(u_i)}{\partial F_u(u_i)} \tag{2}$$

It can be shown that an ICA solution is a stable point of the relaxation of eqs.(1-2). In practical tests on separating mixed speech signals, good results were found when using the logistic function, $y_i = (1 + e^{-u_i})^{-1}$, instead of the known c.d.f., $F_u$, of the speech signals. In this case $\hat{y}_i = 1 - 2y_i$, and the algorithm has a simple form. These results were obtained despite the fact that the p.d.f. of the speech signals was not exactly matched by the gradient of the logistic function. In the experiments in this paper, we also used the speedup technique of prewhitening described in [2].

## 1.3   Applying ICA to EEG Data

The ICA technique appears ideally suited for performing source separation in domains where, (1) the sources are independent, (2) the propagation delays of the 'mixing medium' are negligible, (3) the sources are analog and have p.d.f.'s not too unlike the gradient of a logistic sigmoid, and (4) the number of independent signal sources is the same as the number of sensors, meaning if we employ $N$ sensors, using the ICA algorithm we can separate $N$ sources. In the case of EEG signals, $N$ scalp electrodes pick up correlated signals and we would like to know what effectively 'independent brain sources' generated these mixtures. If we assume that the complexity of EEG dynamics can be modeled, at least in part, as a collection of a modest number of statistically independent brain processes, the EEG source analysis problem satisfies ICA assumption (1). Since volume conduction in brain tissue is effectively instantaneous, ICA assumption (2) is also satisfied. Assumption (3) is plausible, but assumption (4), that the EEG is a linear mixtures of exactly $N$ sources, is questionable, since we do not know the effective number of statistically independent brain signals contributing to the EEG recorded from the scalp. The foremost problem in interpreting the output of ICA is, therefore, determining the proper dimension of input channels, and the physiological and/or psychophysiological significance of the derived ICA source channels.

Although the ICA model of the EEG ignores the known variable synchronization of separate EEG generators by common subcortical or corticocortical influences [5], it appears promising for identifying concurrent signal sources that are either situated too close together, or are too widely distributed to be separated by current localization techniques. Here, we report a first application of the ICA algorithm to analysis of 14-channel EEG and ERP recordings during sustained eyes-closed performance of an auditory detection task, and give evidence suggesting that the ICA algorithm may be useful for identifying psychophysiological state transitions.

# 2   Methods

EEG and behavioral data were collected to develop a method of objectively monitoring the alertness of operators of complex systems [8]. Ten adult volunteers participated in three or more half-hour sessions, during which they pushed one button whenever they detected an above-threshold auditory target stimulus (a brief increase in the level of the continuously-present background noise). To maximize the chance of observing alertness decrements, sessions were conducted in a small, warm, and dimly-lit experimental chamber, and subjects were instructed to keep their eyes closed. Auditory targets were 350 ms increases in the intensity of a 62 dB white noise background, 6 dB above their threshold of detectability, presented at random time intervals at a mean rate of 10/min, and superimposed on a continuous 39-Hz click train evoking a 39-Hz *steady-state response* (SSR). Short, and task-irrelevant probe tones of two frequencies (568 and 1098 Hz) were interspersed between the target noise bursts at 2-4 s intervals. EEG was collected from thirteen electrodes located at sites of the International 10-20 System, referred to the right mastoid, at a sampling rate of 312.5 Hz. A bipolar diagonal electrooculogram (EOG) channel was also recorded for use in eye movement artifact correction and rejection. Target Hits were defined as targets responded to within a 100-3000 ms window, while Lapses were targets not responded to. Two sessions each from three of the subjects were selected for analysis based on their containing at least 50 response Lapses. A continuous performance measure, local error rate, was computed by convolving the irregularly-sampled performance index time series (Hit=0/Lapse=1) with a 95 s smoothing window advanced for 1.64 s steps.

The ICA algorithm in eqs.(1-2) was applied to the 14 EEG recordings. The time index was permuted to ensure signal stationarity, and the 14-dimensional time point vectors were presented to a 14 → 14 ICA network one at a time. To speed convergence, we first pre-whitened the data to remove first- and second-order statistics. The learning rate was annealed from 0.03 to 0.0001 during convergence. After each pass through the whole training set, we checked the amount of correlation between the ICA output channels and the amount of change in weight matrix, and stopped the training procedure when, (1) the mean correlation among all channel pairs was below 0.05, and (2) the ICA weights had stopped changing appreciably.

## 3    Results

A small (4.5 s) portion of the resulting ICA-transformed EEG time series is shown in Figure 1. As expected, correlations between the ICA traces are close to zero. The dominant theta wave (near 7 Hz) spread across many EEG channels (*left panel*) is more or less isolated to ICA trace 1 (*upper right*), both in the epoch shown and throughout the session. Alpha activity (near 10 Hz) not obvious in the EEG data is uncovered in ICA trace 2, which here and throughout the session contains alpha bursts interspersed with quiescent periods. Other ICA traces (3-8) contain brief oscillatory bursts which are not easy to characterize, but clearly display different dynamics from the activity in ICA trace 1 which dominates the raw EEG record. ICA trace 10 contains near-DC changes associated with eye slow movements in the EOG and most frontal (Fpz) EEG channels. ICA trace 13 contains mostly line noise (60 Hz), while ICA traces 9 and 14 have a broader high frequency (50-100 Hz) spectrum, suggesting that their source is likely to be high-frequency activity generated by scalp muscles.

Apparently, the ICA source solution for this data does not depend strongly on learning rate or initial conditions. When the same portion of one session was used to train two ICA networks with different random starting weights, data presentation orders, and learning rates, the two final ICA weight matrices were very close to one another. Filtering another segment of EEG data from the same session using each ICA matrix produced two ICA source transforms in which 11 of the 14 best-correlated output channel pairs correlated above 0.95 and none correlated less than 0.894.

While ICA training minimized mutual information, and therefore also correlations between output channels during the initial (alert) ICA training period, output data channels filtered by the same ICA weight matrix became more correlated during the drowsy portion of the session, and then reverted to their initial levels of (de)correlation when the subject again became alert. Conversely, filtering the same session's data with an ICA weight matrix trained on the drowsy portion of the session produced output channels that were more correlated during the alert portions of the session than during the drowsy training period. Presumably, these changes in residual correlation among ICA outputs reflect changes in the dynamics and topographic structure of the EEG signals in alert and drowsy brain states.

An important problem in human electrophysiology is to determine a means of objectively identifying overlapping ERP subcomponents. Figure 3 (*right panel*) shows an ICA decomposition of (*left panel*) ERPs to detected (Hit) and undetected (Lapse) targets by the same subject. ICA spatial filtering produces two channels (S[1-2]) separating out the 39-Hz *steady-state response* (SSR) produced by the continuous 39-Hz click stimulation during the session. Note the stimulus-induced perturbation in SSR amplitude previously identified in [6]. Three channels (H[1-3]) pass time-limited components of the detected target response, while four others (L[1-4])

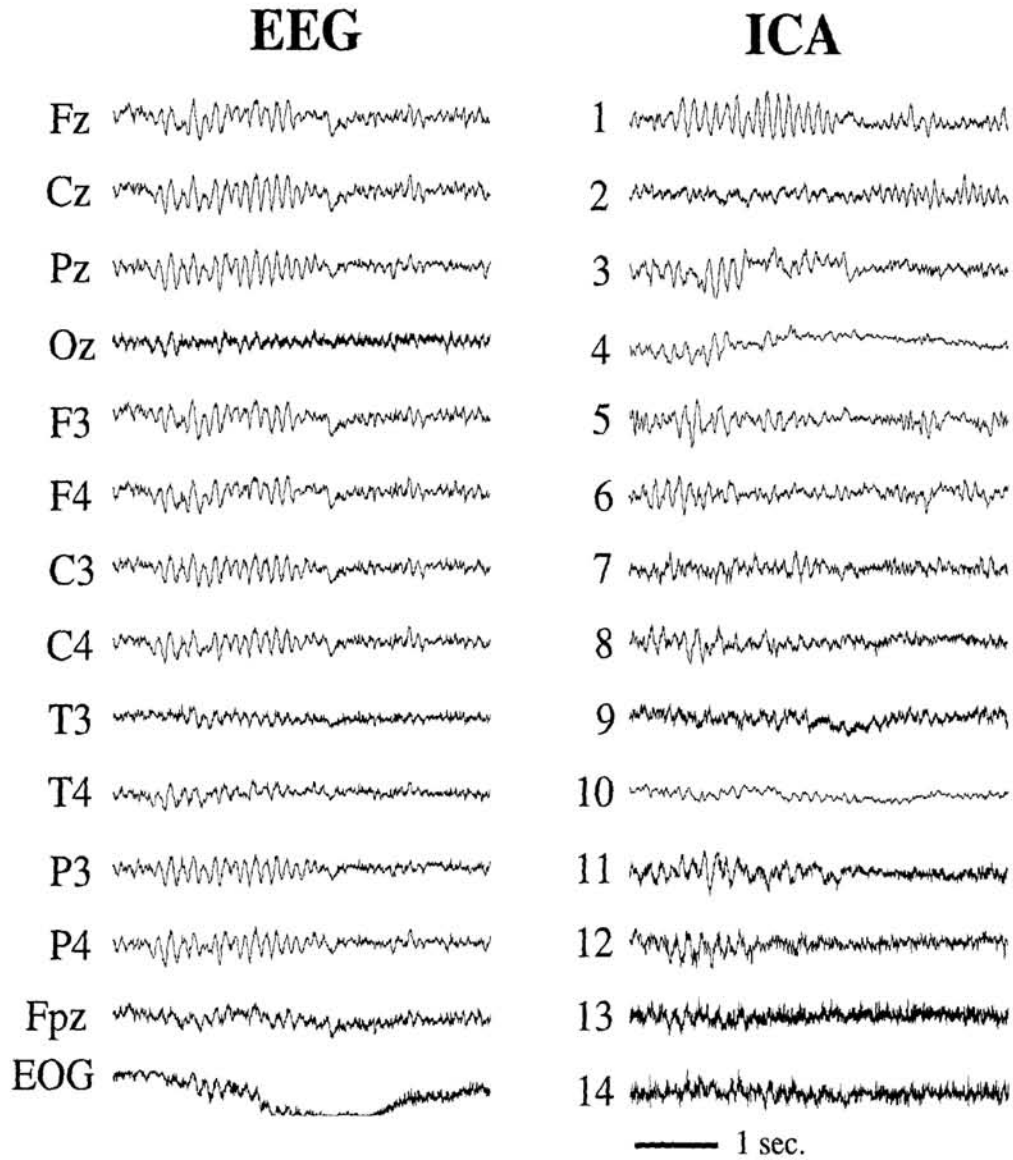

Figure 1: Left: 4.5 seconds of 14-channel EEG data. Right: an ICA transform of the same data, using weights trained on 6.5 minutes of similar data from the same session.

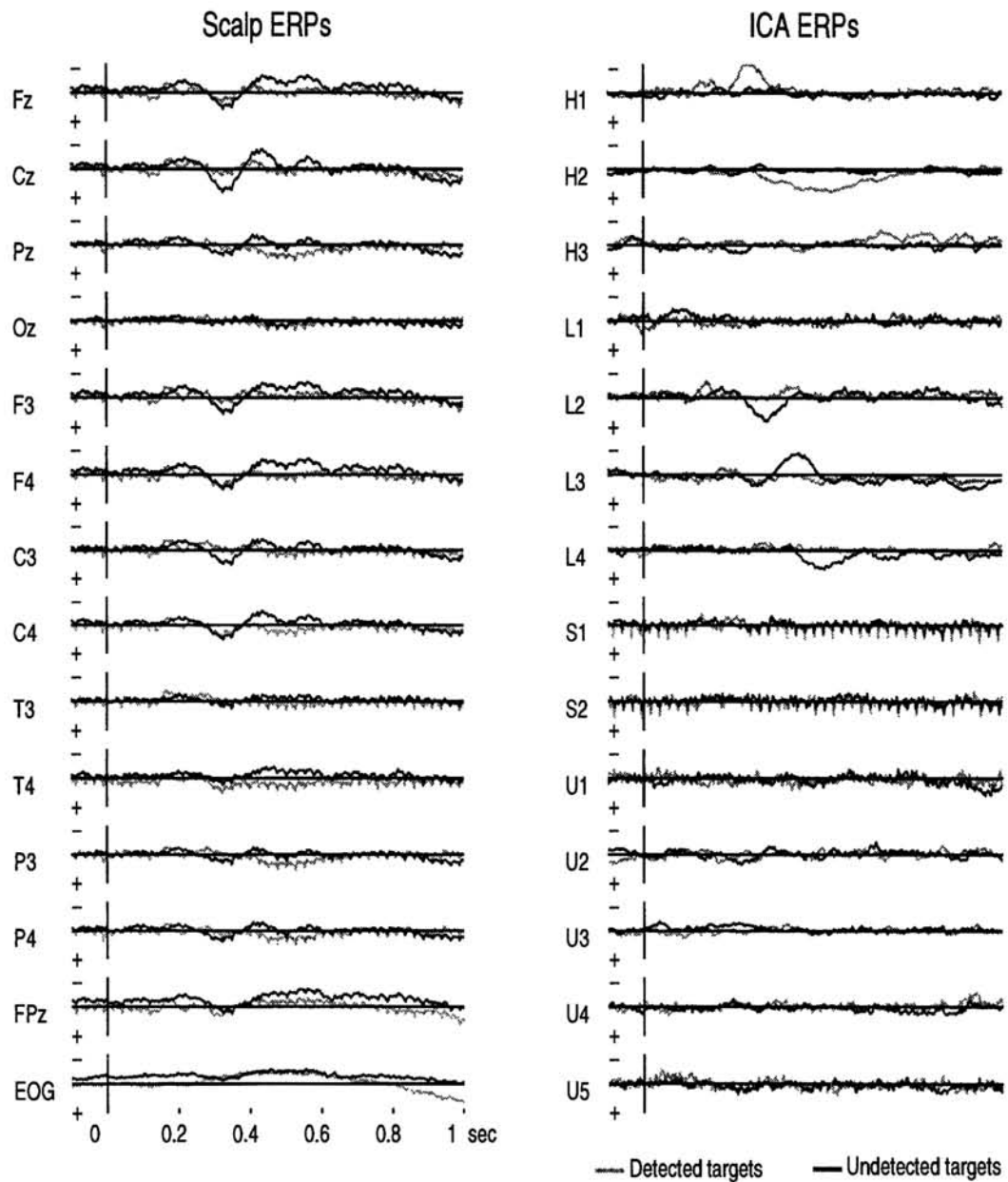

Figure 2: *Left panel*: Event-related potentials (ERPs) in response to undetected (*bold traces*) and detected (*faint traces*) noise targets during two half-hour sessions. *Right panel*: Same ERP signals filtered using an ICA weight matrix trained on the ERP data.

components of the (larger) undetected target response. We suggest these represent the time course of the locus (either focal or distributed) of brain response activity, and may represent a solution to the longstanding problem of objectively dividing evoked responses into neurobiologically meaningful, temporally overlapping sub-components.

## 4   Conclusions

ICA appears to be a promising new analysis tool for human EEG and ERP research. It can isolate a wide range of artifacts to a few output channels while removing them from remaining channels. These may in turn represent the time course of activity in longlasting or transient independent 'brain sources' on which the algorithm converges reliably. By incorporating higher-order statistical information, ICA avoids the non-uniqueness associated with decorrelating decompositions. The algorithm also appears to be useful for decomposing evoked response data into spatially distinct subcomponents, while measures of nonstationarity in the ICA source solution may be useful for observing brain state changes.

## Acknowledgments

This report was supported in part by a grant (ONR.Reimb.30020.6429) to the Naval Health Research Center by the Office of Naval Research. The views expressed in this article are those of the authors and do not reflect the official policy or position of the Department of the Navy, Department of Defense, or the U.S. Government. Dr. Bell is supported by grants from the Office of Naval Research and the Howard Hughes Medical Institute.

## References

[1] A.J. Bell & T.J. Sejnowski (1995). An information-maximization approach to blind separation and blind deconvolution, *Neural Computation* **7**:1129-1159.

[2] A.J. Bell & T.J. Sejnowski (1995). Fast blind separation based on information theory, in *Proc. Intern. Symp. on Nonlinear Theory and Applications (NOLTA)*, Las Vegas, Dec. 1995.

[3] P. Comon (1994) Independent component analysis, a new concept? *Signal processing* **36**:287-314.

[4] A.M. Dale & M.I. Sereno (1993) EEG and MEG source localization: a linear approach. *J. Cogn. Neurosci.* **5**:162.

[5] R. Galambos & S. Makeig. (1989) Dynamic changes in steady-state potentials. In Erol Basar (ed.), *Dynamics of Sensory and Cognitive Processing of the Brain*, 102-122. Berlin:Springer-Verlag.

[6] S. Makeig & R. Galambos. (1989) The CERP: Event-related perturbations in steady-state responses. In E. Basar & T.H. Bullock (ed.), *Brain Dynamics: Progress and Perspectives*, 375-400. Berlin:Springer-Verlag.

[7] T-P. Jung, S. Makeig, M. Stensmo, & T. Sejnowski. Estimating alertness from the EEG power spectrum. Submitted for publication.

[8] S. Makeig & M. Inlow (1993) Lapses in alertness: Coherence of fluctuations in performance and EEG spectrum. *Electroencephalog. clin. Neurophysiolog.* **86**:23-35.